# Active Learning in Multilayer Perceptrons

**Kenji Fukumizu**
Information and Communication R&D Center, Ricoh Co., Ltd.
3-2-3, Shin-yokohama, Yokohama, 222 Japan
E-mail: fuku@ic.rdc.ricoh.co.jp

## Abstract

We propose an active learning method with hidden-unit reduction, which is devised specially for multilayer perceptrons (MLP). First, we review our active learning method, and point out that many Fisher-information-based methods applied to MLP have a critical problem: the information matrix may be singular. To solve this problem, we derive the singularity condition of an information matrix, and propose an active learning technique that is applicable to MLP. Its effectiveness is verified through experiments.

## 1 INTRODUCTION

When one trains a learning machine using a set of data given by the true system, its ability can be improved if one selects the training data actively. In this paper, we consider the problem of active learning in multilayer perceptrons (MLP). First, we review our method of active learning (Fukumizu el al., 1994), in which we prepare a probability distribution and obtain training data as samples from the distribution. This methodology leads us to an information-matrix-based criterion similar to other existing ones (Fedorov, 1972; Pukelsheim, 1993).

Active learning techniques have been recently used with neural networks (MacKay, 1992; Cohn, 1994). Our method, however, as well as many other ones has a crucial problem: the required inverse of an information matrix may not exist (White, 1989).

We propose an active learning technique which is applicable to three-layer perceptrons. Developing a theory on the singularity of a Fisher information matrix, we present an active learning algorithm which keeps the information matrix nonsingular. We demonstrate the effectiveness of the algorithm through experiments.

# 2   STATISTICALLY OPTIMAL TRAINING DATA

## 2.1   A CRITERION OF OPTIMALITY

We review the criterion of statistically optimal training data (Fukumizu *et al.*, 1994). We consider the regression problem in which the target system maps a given input $x$ to $y$ according to

$$y = f(x) + Z,$$

where $f(x)$ is a deterministic function from $\mathbf{R}^L$ to $\mathbf{R}^M$, and $Z$ is a random variable whose law is a normal distribution $N(0, \sigma^2 I_M)$, ($I_M$ is the unit $M \times M$ matrix). Our objective is to estimate the true function $f$ as accurately as possible.

Let $\{f(x; \theta)\}$ be a parametric model for estimation. We use the maximum likelihood estimator (MLE) $\hat{\theta}$ for training data $\{(x^{(\nu)}, y^{(\nu)})\}_{\nu=1}^N$, which minimizes the sum of squared errors in this case. In theoretical derivations, we assume that the target function $f$ is included in the model and equal to $f(\cdot; \theta_0)$.

We make a training example by choosing $x^{(\nu)}$ to try, observing the resulting output $y^{(\nu)}$, and pairing them. The problem of active learning is how to determine input data $\{x^{(\nu)}\}_{\nu=1}^N$ to minimize the estimation error after training. Our approach is a statistical one using a *probability for training*. $r(x)$, and choosing $\{x^{(\nu)}\}_{\nu=1}^N$ as independent samples from $r(x)$ to minimize the expectation of the MSE *in the actual environment*:

$$\text{EMSE} = \text{E}_{\{(x^{(\nu)}, y^{(\nu)})\}} \left[ \int \int \|y - f(x; \hat{\theta})\|^2 p(y|x) dy dQ \right]. \tag{1}$$

In the above equation, $Q$ is *the environmental probability* which gives input vectors to the true system in the actual environment, and $\text{E}_{\{(x^{(\nu)}, y^{(\nu)})\}}$ means the expectation on training data. Eq.(1), therefore, shows the average error of the trained machine that is used as a substitute of the true function in the actual environment.

## 2.2   REVIEW OF AN ACTIVE LEARNING METHOD

Using statistical asymptotic theory, Eq.(1) is approximated as follows:

$$\text{EMSE} = \sigma^2 + \frac{\sigma^2}{N} \text{Tr} \left[ I(\theta_0) J^{-1}(\theta_0) \right] + O(N^{-3/2}), \tag{2}$$

where the matrixes $I$ and $J$ are (*Fisher*) *information matrixes* defined by

$$I_{ab}(x; \theta) = \frac{\partial f^T(x; \theta)}{\partial \theta_a} \frac{\partial f(x; \theta)}{\partial \theta_b}, \quad I(\theta) = \int I(x; \theta) dQ(x). \quad J(\theta) = \int I(x; \theta) r(x) dx.$$

The essential part of Eq.(2) is $\text{Tr}[I(\theta_0) J^{-1}(\theta_0)]$, computed by the unavailable parameter $\theta_0$. We have proposed a practical algorithm in which we replace $\theta_0$ with $\hat{\theta}$. prepare a family of probability $\{r(x; v) \mid v : \text{paramater}\}$ to choose training samples, and optimize $v$ and $\hat{\theta}$ iteratively (Fukumizu *et al.*, 1994).

**Active Learning Algorithm**

  1. Select an initial training data set $D_{[0]}$ from $r(x; v_{[0]})$. and compute $\hat{\theta}_{[0]}$.

  2. $k := 1$.

  3. Compute the optimal $v = v_{[k]}$ to minimize $\text{Tr}[I(\hat{\theta}_{[k-1]}) J^{-1}(\hat{\theta}_{[k-1]})]$.

4. Choose $N_k$ new training data from $r(x; v_{[k]})$ and let $D_{[k]}$ be a union of $D_{[k-1]}$ and the new data.

5. Compute the MLE $\hat{\theta}_{[k]}$ based on the training data set $D_{[k]}$.

6. $k := k + 1$ and go to 3.

The above method utilizes a probability to generate training data. It has the advantage of making many data in one step compared to existing ones in which only one data is chosen in each step, though their criterions are similar to each other.

## 3   SINGULARITY OF AN INFORMATION MATRIX

### 3.1   A PROBLEM ON ACTIVE LEARNING IN MLP

Hereafter, we focus on active learning in three-layer perceptrons with H hidden units, $\mathcal{N}_H = \{f(x, \theta)\}$. The map $f(x; \theta)$ is defined by

$$f_i(x; \theta) = \sum_{j=1}^{H} w_{ij}\, s(\sum_{k=1}^{L} u_{jk} x_k + \zeta_j) + \eta_i, \qquad (1 \le i \le M), \tag{3}$$

where $s(t)$ is the sigmoidal function: $s(t) = 1/(1 + e^{-t})$.

Our active learning method as well as many other ones requires the inverse of an information matrix $J$. The information matrix of MLP, however, is not always invertible (White, 1989). Any statistical algorithms utilizing the inverse, then, cannot be applied directly to MLP (Hagiwara et al., 1993). Such problems do not arise in linear models, which almost always have a nonsingular information matrix.

### 3.2   SINGULARITY OF AN INFORMATION MATRIX OF MLP

The following theorem shows that the information matrix of a three-layer perceptron is singular if and only if the network has redundant hidden units. We can deduce that if the information matrix is singular, we can make it nonsingular by eliminating redundant hidden units without changing the input-output map.

**Theorem 1** *Assume $r(x)$ is continuous and positive at any $x$. Then, the Fisher information matrix $J$ is singular if and only if at least one of the following three conditions is satisfied:*
*(1) $u_j := (u_{j1}, \ldots, u_{jL})^T = 0$, for some $j$.*
*(2) $w_j := (w_{1j}, \ldots, w_{Mj}) = 0^T$, for some $j$.*
*(3) For different $j_1$ and $j_2$, $(u_{j_1}^T, \zeta_{j_1}) = (u_{j_2}^T, \zeta_{j_2})$ or $(u_{j_1}^T, \zeta_{j_1}) = -(u_{j_2}^T, \zeta_{j_2})$.*

The rough sketch of the proof is shown below. The complete proof will appear in a forthcoming paper (Fukumizu, 1996).
*Rough sketch of the proof.*   We know easily that an information matrix is singular if and only if $\{\frac{\partial f(x;\theta)}{\partial \theta_a}\}_a$ are linearly dependent. The sufficiency can be proved easily. To show the necessity, we show that the derivatives are linearly independent if none of the three conditions is satisfied. Assume a linear relation:

$$\sum_{i=1}^{M}\sum_{j=1}^{H} \alpha_{ij} \frac{\partial f}{\partial w_{ij}} + \sum_{i=1}^{M} \alpha_{i0} \frac{\partial f}{\partial \eta_i} + \sum_{j=1}^{H}\sum_{k=1}^{L} \beta_{jk} \frac{\partial f}{\partial u_{jk}} + \sum_{j=1}^{H} \beta_{j0} \frac{\partial f}{\partial \zeta_j} = o. \tag{4}$$

We can show there exists a basis of $\mathbf{R}^L$, $\langle \boldsymbol{x}^{(1)}, \ldots, \boldsymbol{x}^{(L)} \rangle$, such that $\boldsymbol{u}_j \cdot \boldsymbol{x}^{(l)} \neq 0$ for $\forall j$, $\forall l$, and $\boldsymbol{u}_{j_1} \cdot \boldsymbol{x}^{(l)} + \zeta_{j_1} \neq \pm(\boldsymbol{u}_{j_2} \cdot \boldsymbol{x}^{(l)} + \zeta_{j_2})$ for $j_1 \neq j_2, \forall l$. We replace $\boldsymbol{x}$ in eq.(4) by $\boldsymbol{x}^{(l)} t$ $(t \in \mathbf{R})$. Let $m_j^{(l)} := \boldsymbol{u}_j \cdot \boldsymbol{x}^{(l)}$, $S_j^{(l)} := \{z \in \mathbf{C} \mid z = ((2n+1)\pi\sqrt{-1} - \zeta_j)/m_j^{(l)}$, $n \in \mathbf{Z}\}$, and $D^{(l)} := \mathbf{C} - \cup_j S_j^{(l)}$. The points in $S_j^{(l)}$ are the singularities of $s(m_j^{(l)} z + \zeta_j)$. We define holomorphic functions on $D^{(l)}$ as

$$
\begin{aligned}
\Psi_i^{(l)}(z) \; := \; & \textstyle\sum_{j=1}^H \alpha_{ij} s(m_j^{(l)} z + \zeta_j) + \alpha_{i0} + \sum_{j=1}^H \sum_{k=1}^L \beta_{jk} w_{ij} s'(m_j^{(l)} z + \zeta_j) x_k^{(l)} z \\
& + \textstyle\sum_{j=1}^H \beta_{j0} w_{ij} s'(m_j^{(l)} z + \zeta_j), \qquad\qquad (1 \le i \le M).
\end{aligned}
$$

From eq.(4), we have $\Psi_i^{(l)}(t) = 0$ for all $t \in \mathbf{R}$. Using standard arguments on isolated singularities of holomorphic functions, we know $S_j^{(l)}$ are removable singularities of $\Psi_i^{(l)}(z)$, and finally obtain

$$
w_{ij} \textstyle\sum_{k=1}^L \beta_{jk} x_k^{(l)} = 0, \qquad w_{ij} \beta_{j0} = 0, \qquad \alpha_{ij} = 0, \qquad \alpha_{i0} = 0.
$$

It is easy to see $\beta_{jk} = 0$. This completes the proof.

## 3.3  REDUCTION PROCEDURE

We introduce the following *reduction procedure* based on Theorem 1. Used during BP training, it eliminates redundant hidden units and keeps the information matrix nonsingular. The criterion of elimination is very important, because excessive elimination of hidden units degrades the approximation capacity. We propose an algorithm which does not increase the mean squared error on average. In the following, let $\hat{s}_j := s(\hat{\boldsymbol{u}}_j \cdot \boldsymbol{x} + \hat{\eta}_j)$ and $\varepsilon(N) = A/N$ for a positive number $A$.

**Reduction Procedure**

1. If    $\|\hat{\boldsymbol{w}}_j\|^2 \int (\hat{s}_j - s(\hat{\zeta}_j))^2 dQ < \varepsilon(N)$,    then eliminate the $j$th hidden unit, and $\hat{\eta}_i \to \hat{\eta}_i + \hat{w}_{ij} s(\hat{\zeta}_j)$ for all $i$.

2. If    $\|\hat{\boldsymbol{w}}_j\|^2 \int (\hat{s}_j)^2 dQ < \varepsilon(N)$,    then eliminate the $j$th hidden unit.

3. If    $\|\hat{\boldsymbol{w}}_{j_2}\|^2 \int (\hat{s}_{j_2} - \hat{s}_{j_1})^2 dQ < \varepsilon(N)$    for different $j_1$ and $j_2$, then eliminate the $j_2$th hidden unit and $\hat{w}_{ij_1} \to \hat{w}_{ij_1} + \hat{w}_{ij_2}$ for all $i$.

4. If    $\|\hat{\boldsymbol{w}}_{j_2}\|^2 \int (1 - \hat{s}_{j_2} - \hat{s}_{j_1})^2 dQ < \varepsilon(N)$    for different $j_1$ and $j_2$, then eliminate the $j_2$th hidden unit and $\hat{w}_{ij_1} \to \hat{w}_{ij_1} - \hat{w}_{ij_2}$,   $\hat{\eta}_i \to \hat{\eta}_i + \hat{w}_{ij_2}$   for all $i$.

From Theorem 1, we know that $\hat{\boldsymbol{w}}_j$, $\hat{\boldsymbol{u}}_j$, $(\hat{\boldsymbol{u}}_{j_2}^T, \hat{\zeta}_{j_2}) - (\hat{\boldsymbol{u}}_{j_1}^T, \hat{\zeta}_{j_1})$, or $(\hat{\boldsymbol{u}}_{j_2}^T, \hat{\zeta}_{j_2}) + (\hat{\boldsymbol{u}}_{j_1}^T, \hat{\zeta}_{j_1})$ can be reduced to 0 if the information matrix is singular. Let $\tilde{\theta} \in \mathcal{N}_K$ denote the reduced parameter from $\hat{\theta}$ according to the above procedure. The above four conditions are, then, given by calculating $\int \|\boldsymbol{f}(x; \tilde{\theta}) - \boldsymbol{f}(x; \hat{\theta})\|^2 dQ$.

We briefly explain how the procedure keeps the information matrix nonsingular and does not increase EMSE in high probability. First, suppose $\det J(\theta_0) = 0$, then there exists $\theta_0^K \in \mathcal{N}_K$ $(K < H)$ such that $f(x; \theta_0) = f(x; \theta_0^K)$ and $\det J(\theta_0^K) \neq 0$ in $\mathcal{N}_K$. The elimination of hidden units up to $K$, of course, does not increase the EMSE. Therefore, we have only to consider the case in which $\det J(\theta_0) \neq 0$ and hidden units are eliminated.

Suppose $\int \|\boldsymbol{f}(\boldsymbol{x}; \theta_0^K) - \boldsymbol{f}(\boldsymbol{x}; \theta_0)\|^2 dQ > O(N^{-1})$ for any reduced parameter $\theta_0^K$ from $\theta_0$. The probability of satisfying $\int \|\boldsymbol{f}(\boldsymbol{x}; \tilde{\theta}) - \boldsymbol{f}(\boldsymbol{x}; \hat{\theta})\|^2 dQ < A/N$ is very small for

a sufficiently small $A$. Thus, the elimination of hidden units occurs in very tiny probability. Next, suppose $\int \|f(x;\theta_0^K) - f(x;\theta_0)\|^2 dQ = O(N^{-1})$. Let $\tilde{\theta} \in \mathcal{N}_K$ be a reduced parameter made from $\hat{\theta}$ with the same procedure as we obtain $\theta_0^K$ from $\theta_0$. We will show for a sufficiently small $A$,

$$\mathrm{E}\left[\int \|f(x;\hat{\theta}^K) - f(x;\theta_0)\|^2 dQ\right] \leq \mathrm{E}\left[\int \|f(x;\hat{\theta}) - f(x;\theta_0)\|^2 dQ\right],$$

where $\hat{\theta}^K$ is MLE computed in $\mathcal{N}_K$. We write $\theta = (\theta^{(1)}, \theta^{(2)})$ in which $\theta^{(2)}$ is changed to 0 in reduction, changing the coordinate system if necessary. The Taylor expansion and asymptotic theory give

$$\mathrm{E}\left[\int \|f(x;\hat{\theta}^K) - f(x;\theta_0)\|^2 dQ\right] \approx \int \|f(x;\theta_0^K) - f(x;\theta_0)\|^2 dQ + \frac{\sigma^2}{N}\mathrm{Tr}[I_{11}(\theta_0^K)J_{11}^{-1}(\theta_0^K)],$$

$$\mathrm{E}\left[\int \|f(x;\tilde{\theta}) - f(x;\hat{\theta})\|^2 dQ\right] \approx \int \|f(x;\theta_0^K) - f(x;\theta_0)\|^2 dQ + \frac{\sigma^2}{N}\mathrm{Tr}[I_{22}(\theta_0^K)J_{22}^{-1}(\theta_0)],$$

where $I_{ii}$ and $J_{ii}$ denote the local information matrixes w.r.t. $\theta^{(i)}$ ($i = 1, 2$). Thus,

$$\begin{aligned} \mathrm{E}&\left[\int \|f(x;\hat{\theta}) - f(x;\theta_0)\|^2 dQ\right] - \mathrm{E}\left[\int \|f(x;\hat{\theta}^K) - f(x;\theta_0)\|^2 dQ\right] \\ &\approx \quad -\mathrm{E}\left[\int \|f(x;\tilde{\theta}) - f(x;\hat{\theta})\|^2 dQ\right] + \frac{\sigma^2}{N}\mathrm{Tr}[I_{22}(\theta_0^K)J_{22}^{-1}(\theta_0)] \\ &\quad -\frac{\sigma^2}{N}\mathrm{Tr}[I_{11}(\theta_0^K)J_{11}^{-1}(\theta_0^K)] + \mathrm{E}\left[\int \|f(x;\hat{\theta}) - f(x;\theta_0)\|^2 dQ\right]. \end{aligned}$$

Since the sum of the last two terms is positive, the l.h.s is positive if $\mathrm{E}[\int \|f(x;\hat{\theta}^K) - f(x;\hat{\theta})\|^2 dQ] < B/N$ for a sufficiently small $B$. Although we cannot know the value of this expectation, we can make the probability of holding this enequality very high by taking a small $A$.

## 4 ACTIVE LEARNING WITH REDUCTION PROCEDURE

The reduction procedure keeps the information matrix nonsingular and makes the active learning algorithm applicable to MLP even with surplus hidden units.

**Active Learning with Hidden Unit Reduction**

1. Select initial training data set $D_0$ from $r(x; v_{[0]})$. and compute $\hat{\theta}_{[0]}$.

2. $k := 1$, and do REDUCTION PROCEDURE.

3. Compute the optimal $v = v_{[k]}$ to minimize $\mathrm{Tr}[I(\hat{\theta}_{[k-1]})J^{-1}(\hat{\theta}_{[k-1]})]$. using the steepest descent method.

4. Choose $N_k$ new training data from $r(x; v_{[k]})$ and let $D_{[k]}$ be a union of $D_{[k-1]}$ and the new data.

5. Compute the MLE $\hat{\theta}_{[k]}$ based on the training data $D_{[k]}$ using BP with REDUCTION PROCEDURE.

6. $k := k + 1$ and go to 3.

The BP with reduction procedure is applicable not only to active learning, but to a variety of statistical techniques that require the inverse of an information matrix. We do not discuss it in this paper. however.

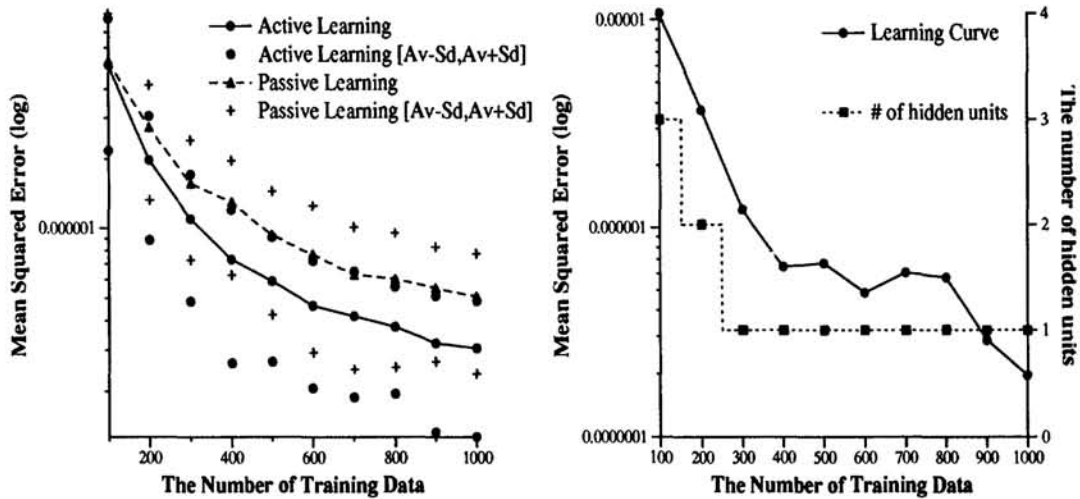

Figure 1: Active/Passive Learning : $f(x) = s(x)$

## 5  EXPERIMENTS

We demonstrate the effect of the proposed active learning algorithm through experiments. First we use a three-layer model with 1 input unit, 3 hidden units, and 1 output unit. The true function $f$ is a MLP network with 1 hidden unit. The information matrix is singular at $\theta_0$, then. The environmental probability, $Q$, is a normal distribution $N(0, 4)$. We evaluate the generalization error in the actual environment using the following mean squared error of the function values:

$$\int \|f(x; \hat{\theta}) - f(x)\|^2 dQ.$$

We set the deviation in the true system $\sigma = 0.01$. As a family of distributions for training $\{r(x; v)\}$, a mixture model of 4 normal distributions is used. In each step of active learning, 100 new samples are added. A network is trained using online BP, presented with all training data 10000 times in each step, and operated the reduction procedure once a 100 cycles between 5000th and 10000th cycle. We try 30 trainings changing the seed of random numbers. In comparison, we train a network passively based on training samples given by the probability $Q$.

Fig.1 shows the averaged learning curves of active/passive learning and the number of hidden units in a typical learning curve. The advantage of the proposed active learning algorithm is clear. We can find that the algorithm has expected effects on a simple, ideal approximation problem.

Second, we apply the algorithm to a problem in which the true function is not included in the MLP model. We use MLP with 4 input units, 7 hidden units, and 1 output unit. The true function is given by $f(x) = \text{erf}(x_1)$, where $\text{erf}(t)$ is the *error function*. The graph of the error function resembles that of the sigmoidal function, while they never coincide by any affine transforms. We set $Q = N(0, 25 \times I_4)$. We train a network actively/passively based on 10 data sets, and evaluate MSE's of function values. Other conditions are the same as those of the first experiment.

Fig.2 shows the averaged learning curves and the number of hidden units in a typical learning curve. We find that the active learning algorithm reduces the errors though the theoretical condition is not perfectly satisfied in this case. It suggests the robustness of our active learning algorithm.

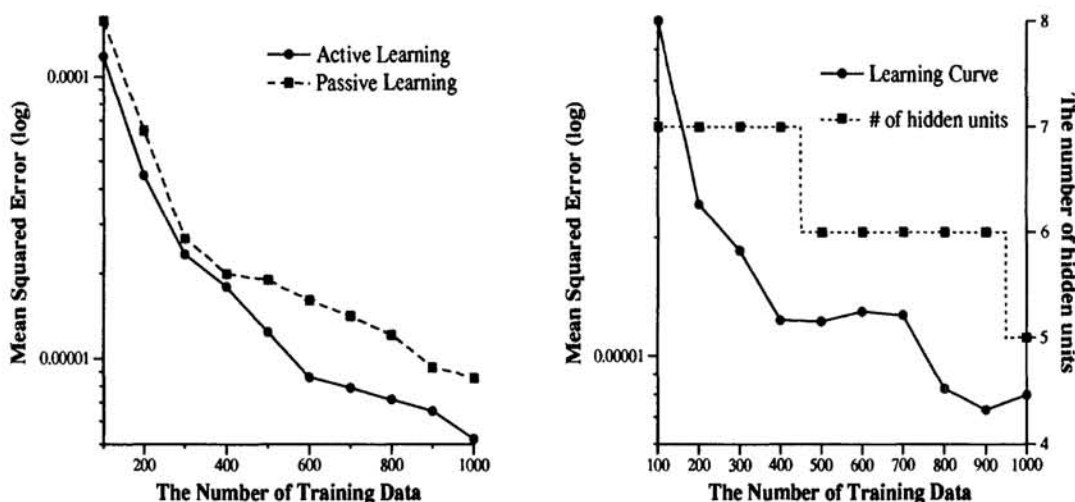

Figure 2: Active/Passive Learning : $f(x) = \mathrm{erf}(x_1)$

## 6  CONCLUSION

We review statistical active learning methods and point out a problem in their application to MLP: the required inverse of an information matrix does not exist if the network has redundant hidden units. We characterize the singularity condition of an information matrix and propose an active learning algorithm which is applicable to MLP with any number of hidden units. The effectiveness of the algorithm is verified through computer simulations, even when the theoretical assumptions are not perfectly satisfied.

### References

D. A. Cohn. (1994) Neural network exploration using optimal experiment design. In J. Cowan et al. (ed.), *Advances in Neural Information Processing Systems 6*, 679-686. San Mateo, CA: Morgan Kaufmann.

V. V. Fedorov. (1972) *Theory of Optimal Experiments*. NY: Academic Press.

K. Fukumizu. (1996) A Regularity Condition of the Information Matrix of a Multilayer Perceptron Network. *Neural Networks*, to appear.

K. Fukumizu, & S. Watanabe. (1994) Error Estimation and Learning Data Arrangement for Neural Networks. *Proc. IEEE Int. Conf. Neural Networks* :777-780.

K. Hagiwara, N. Toda, & S. Usui. (1993) On the problem of applying AIC to determine the structure of a layered feed-forward neural network. *Proc. 1993 Int. Joint Conf. Neural Networks* :2263-2266.

D. MacKay. (1992) Information-based objective functions for active data selection, *Neural Computation* 4(4):305-318.

F. Pukelsheim. (1993) *Optimal Design of Experiments*. NY: John Wiley & Sons.

H. White. (1989) Learning in artificial neural networks: A statistical perspective *Neural Computation* 1(4):425-464.
